# Non-stationary dynamic Bayesian networks

**Joshua W. Robinson and Alexander J. Hartemink**
Department of Computer Science
Duke University
Durham, NC 27708-0129
{josh,amink}@cs.duke.edu

## Abstract

A principled mechanism for identifying conditional dependencies in time-series data is provided through structure learning of dynamic Bayesian networks (DBNs). An important assumption of DBN structure learning is that the data are generated by a stationary process—an assumption that is not true in many important settings. In this paper, we introduce a new class of graphical models called non-stationary dynamic Bayesian networks, in which the conditional dependence structure of the underlying data-generation process is permitted to change over time. Non-stationary dynamic Bayesian networks represent a new framework for studying problems in which the structure of a network is evolving over time. We define the non-stationary DBN model, present an MCMC sampling algorithm for learning the structure of the model from time-series data under different assumptions, and demonstrate the effectiveness of the algorithm on both simulated and biological data.

## 1   Introduction

Structure learning of dynamic Bayesian networks allows conditional dependencies to be identified in time-series data with the assumption that the data are generated by a distribution that does not change with time (i.e., it is stationary). An assumption of stationarity is adequate in many situations since certain aspects of data acquisition or generation can be easily controlled and repeated. However, other interesting and important circumstances exist where that assumption does not hold and potential non-stationarity cannot be ignored.

As one example, structure learning of DBNs has been used widely in reconstructing transcriptional regulatory networks from gene expression data [1]. But during development, these regulatory networks are evolving over time, with certain conditional dependencies between gene products being created as the organism develops, while others are destroyed. As another example, dynamic Bayesian networks have been used to identify the networks of neural information flow that operate in the brains of songbirds [2]. However, as the songbird learns from its environment, the networks of neural information flow are themselves slowly adapting to make the processing of sensory information more efficient. As yet another example, one can use a DBN to model traffic flow patterns. The roads upon which traffic passes do not change on a daily basis, but the dynamic utilization of those roads changes daily during morning rush, lunch, evening rush, and weekends.

If one collects time-series data describing the levels of gene products in the case of transcriptional regulation, neural activity in the case of neural information flow, or traffic density in the case of traffic flow, and attempts to learn a DBN describing the conditional dependencies in these time-series, one could be seriously misled if the data-generation process is non-stationary.

Here, we introduce a new class of graphical model called a non-stationary dynamic Bayesian network (nsDBN), in which the conditional dependence structure of the underlying data-generation

process is permitted to change over time. In the remainder of the paper, we introduce and define the nsDBN framework, present a simple but elegant algorithm for efficiently learning the structure of an nsDBN from time-series data under different assumptions, and demonstrate the effectiveness of these algorithms on both simulated and experimental data.

## 1.1 Previous work

In this paper, we are interested in identifying how the conditional dependencies between time-series change over time; thus, we focus on the task of inferring network structure as opposed to parameters of the graphical model. In particular, we are not as interested in making predictions about future data (such as spam prediction via a naïve Bayes classifier) as we are in analysis of collected data to identify non-stationary relationships between variables in multivariate time-series. Here we describe the few previous approaches to identifying non-stationary networks and discuss the advantages and disadvantages of each. The model we describe in this paper has none of the disadvantages of the models described below primarily because it makes fewer assumptions about the relationships between variables.

Recent work modeling the temporal progression of networks from the social networks community includes an extension to the discrete temporal network model [3], in which the the networks are latent (unobserved) variables that generate observed time-series data [4]. Unfortunately, this technique has certain drawbacks: the variable correlations remain constant over time, only undirected edges can be identified, and segment or epoch divisions must be identified *a priori*.

In the continuous domain, some research has focused on learning the structure of a time-varying Gaussian graphical model [5] with a reversible-jump MCMC approach to estimate the time-varying variance structure of the data. However, some limitations of this method include: the network evolution is restricted to changing at most a single edge at a time and the total number of segments is assumed known *a priori*. A similar algorithm—also based on Gaussian graphical models—iterates between a convex optimization for determining the graph structure and a dynamic programming algorithm for calculating the segmentation [6]. This approach is fast, has no single edge change restriction, and the number of segments is calculated *a posteriori*; however, it does require that the graph structure is decomposable. Additionally, both of the aforementioned approaches only identify undirected edges and assume that the networks in each segment are independent, preventing data and parameters from being shared between segments.

## 2 Brief review of structure learning of Bayesian networks

Bayesian networks are directed acyclic graphical models that represent conditional dependencies between variables as edges. They define a simple decomposition of the complete joint distribution— a variable is conditionally independent of its non-descendants given its parents. Therefore, the joint distribution of every variable $x_i$ can be rewritten as $\prod_i P(x_i | \pi_i, \theta_i)$, where $\pi_i$ are the parents of $x_i$, and $\theta_i$ parameterizes the conditional probability distribution between a variable and its parents. The posterior probability of a given network $G$ (i.e., the set of conditional dependencies) after having observed data $D$ is estimated via Bayes' rule: $P(G|D) \propto P(D|G)P(G)$. The structure prior $P(G)$ can be used to incorporate prior knowledge about the network structure, either about the existence of specific edges or the topology more generally (e.g., sparse); if prior information is not available, this is often assumed uniform. The marginal likelihood $P(D|G)$ can be computed exactly, given a conjugate prior for $\theta_i$. When the $\theta_i$ are independent and multinomially distributed, a Dirichlet conjugate prior is used, and the data are complete, the exactly solution for the marginal likelihood is the Bayesian-Dirichlet equivalent (BDe) metric [7]. Since we will be modifying it later in this paper, we show the expression for the BDe metric here:

$$P(D|G) = \prod_{i=1}^{n} \prod_{j=1}^{q_i} \frac{\Gamma(\alpha_{ij})}{\Gamma(\alpha_{ij} + N_{ij})} \prod_{k=1}^{r_i} \frac{\Gamma(\alpha_{ijk} + N_{ijk})}{\Gamma(\alpha_{ijk})} \tag{1}$$

where $q_i$ is the number of configurations of the parent set $\pi_i$, $r_i$ is the number of discrete states of variable $x_i$, $N_{ij} = \sum_{k=1}^{r_i} N_{ijk}$, $N_{ijk}$ is the number of times $X_i$ took on the value $k$ given the parent configuration $j$, and $\alpha_{ij}$ and $\alpha_{ijk}$ are Dirichlet hyper-parameters on various entries in $\Theta$. If $\alpha_{ijk}$ is set everywhere to $\alpha/(q_i r_i)$, we get a special case of the BDe metric: the uniform BDe metric (BDeu).

Given a metric for evaluating the marginal likelihood $P(D|G)$, a technique for finding the best network(s) must be chosen. Heuristic search methods (i.e., simulated annealing, greedy hill-climbing) may be used to find a best network or set of networks. Alternatively, sampling methods may be used to estimate a posterior over all networks [8]. If the best network is all that is desired, heuristic searches will typically find it more quickly than sampling techniques. In settings where many modes are expected, sampling techniques will more accurately capture posterior probabilities regarding various properties of the network.

Finally, once a search or sampling strategy has been selected, we must determine how to move through the space of all networks. A *move set* defines a set of local traversal operators for moving from a particular state (i.e., a network) to nearby states. Ideally, the move set includes changes that allow posterior modes to be frequently visited. For example, it is reasonable to assume that networks that differ by a single edge will have similar likelihoods. A well designed move set results in fast convergence since less time is spent in the low probability regions of the state space. For Bayesian networks, the move set is often chosen to be {*add an edge*, *delete an edge*, and *reverse an edge*} [8].

DBNs are an extension of Bayesian networks to time-series data, enabling cyclic dependencies between variables to be modeled across time. Structure learning of DBNs is essentially the same as described above, except that modeling assumptions are made regarding how far back in time one variable can depend on another (minimum and maximum lag), and constraints need to be placed on edges so that they do not go backwards in time. For notational simplicity, we assume hereafter that the minimum and maximum lag are both 1. More detailed reviews of structure learning can be found in [9, 10].

## 3   Learning non-stationary dynamic Bayesian networks

We would like to extend the dynamic Bayesian network model to account for non-stationarity. In this section, we detail how the structure learning procedure for DBNs to must be changed to account for non-stationarity when learning non-stationary DBNs (nsDBNs).

Assume that we observe the state of $n$ random variables at $N$ discrete times. Call this multivariate time-series data $D$, and further assume that it is generated according to a non-stationary process, which is unknown. The process is non-stationary in the sense that the network of conditional dependencies prevailing at any given time is itself changing over time. We call the initial network of conditional dependencies $G_1$ and subsequent networks are called $G_i$ for $i = 2, 3, \ldots, m$. We define $\Delta g_i$ to be the set of edges that change (either added or deleted) between $G_i$ and $G_{i+1}$. The number of edge changes specified in $\Delta g_i$ is $S_i$. We define the *transition time* $t_i$ to be the time at which $G_i$ is replaced by $G_{i+1}$ in the data-generation process. We call the period of time between consecutive transition times—during which a single network of conditional dependencies is operative—an *epoch*. So we say that $G_1$ prevails during the first epoch, $G_2$ prevails during the second epoch, and so forth. We will refer to the entire series of prevailing networks as the *structure* of the nsDBN.

Since we wish to learn a set of networks instead of one network we must derive a new expression for the marginal likelihood. Assume that there exist $m$ different epochs with $m - 1$ transition times $T = \{t_1, \ldots, t_{m-1}\}$. The network $G_{i+1}$ prevailing in epoch $i + 1$ differs from network $G_i$ prevailing in epoch $i$ by a set of edge changes we call $\Delta g_i$. We would like to determine the sequence of networks $G_1, \ldots, G_m$ that maximize the posterior:

$$
\begin{aligned}
P(G_1, \ldots, G_m | D, T) \quad &\propto \quad P(D|G_1, \ldots, G_m, T)P(G_1, \ldots, G_m) &\quad (2)\\
&\propto \quad P(D|G_1, \Delta g_1, \ldots, \Delta g_{m-1}, T)P(G_1, \Delta g_1, \ldots, \Delta g_{m-1}) &\quad (3)\\
&\propto \quad P(D|G_1, \Delta g_1, \ldots, \Delta g_{m-1}, T)P(G_1)P(\Delta g_1, \ldots, \Delta g_{m-1}) &\quad (4)
\end{aligned}
$$

We assume the prior over networks can be further split into independent components describing the initial network and subsequent edge changes, as demonstrated in Equation (4). As in the stationary setting, if prior knowledge about particular edges or overall topology is available, an informative prior can be placed on $G_1$. In the results reported here, we assume this to be uniform. We do, however, place some prior assumptions on the ways in which edges change in the structure. First, we assume that the networks evolve smoothly over time. To encode this prior knowledge, we place an exponential prior with rate $\lambda_s$ on the total number of edge changes $s = \sum_i S_i$. We also assume that the networks evolve slowly over time (i.e., a transition does not occur at every observation) by

placing another exponential prior with rate $\lambda_m$ on the number of epochs $m$. The updated posterior for an nsDBN structure is given as:

$$P(G_1, \Delta g_1, \ldots, \Delta g_{m-1} | T) \propto P(D | G_1, \Delta g_1, \ldots, \Delta g_{m-1}, T) \, e^{-\lambda_s s} e^{-\lambda_m m}$$

To evaluate the new likelihood, we choose to extend the BDe metric because after the parameters have been marginalized away, edges are the only representation of conditional dependencies that are left; this provides a useful definition of non-stationarity that is both simple to define and easy to analyze. We will assume that any other sources of non-stationarity are either small enough to not alter edges in the predicted network or large enough to be approximated by edge changes in the predicted network.

In Equation (1), $N_{ij}$ and $N_{ijk}$ are calculated for a particular parent set over the entire dataset $D$. However, in an nsDBN, a node may have multiple parent sets operative at different times. The calculation for $N_{ij}$ and $N_{ijk}$ must therefore be modified to specify the intervals during which each parent set is operative. Note that an interval may be defined over several epochs. Specifically, an epoch is defined between adjacent transition times while an interval is defined over the epochs during which a particular parent set is operative (which may include all epochs).

For each node $i$, the previous parent set $\pi_i$ in the BDe metric is replaced by a set of parent sets $\pi_{ih}$, where $h$ indexes the interval $I_h$ during which parent set $\pi_{ih}$ is operative for node $i$. Let $p_i$ be the number of such intervals and let $q_{ih}$ be the number of configurations of $\pi_{ih}$. Then we can write:

$$P(D | G_1, \ldots, G_m, T) \propto \prod_{i=1}^{n} \prod_{h=1}^{p_i} \prod_{j=1}^{q_{ih}} \frac{\Gamma(\alpha_{ij}(I_h))}{\Gamma(\alpha_{ij}(I_h) + N_{ij}(I_h))} \prod_{k=1}^{r_i} \frac{\Gamma(\alpha_{ijk}(I_h) + N_{ijk}(I_h))}{\Gamma(\alpha_{ijk}(I_h))} \qquad (5)$$

where the counts $N_{ijk}$ and pseudocounts $\alpha_{ijk}$ have been modified to apply only to the data in each interval $I_h$. The modified BDe metric will be referred to as nsBDe. We have chosen to set $\alpha_{ijk}(I_h) = (\alpha_{ijk}|I_h|)/N$ (e.g., proportional to the length of the interval during which that particular parent set is operative).

We use a sampling approach rather than heuristic search because the posterior over structures includes many modes. Additionally, sampling allows us to answer questions like "what are the most likely transition times?"—a question that would be difficult to answer in the context of heuristic search.

Because the number of possible nsDBN structures is so large (significantly greater than the number of possible DBNs), we must be careful about what options are included in the move set. To achieve quick convergence, we want to ensure that every move in the move set efficiently jumps between posterior modes. Therefore, the majority of the next section is devoted to describing effective move sets under different levels of uncertainty.

## 4   Different settings regarding the number and times of transitions

An nsDBN can be identified under a variety of *settings* that differ in the level of uncertainty about the number of transitions and whether the transition times are known. The different settings are abbreviated according to the type of uncertainty: whether the number of transitions is known (KN) or unknown (UN) and whether the transition times themselves are known (KT) or unknown (UT).

When the number and times of transitions are known *a priori* (KNKT setting), we only need to identify the most likely initial network $G_1$ and sets of edge changes $\Delta g_1 \ldots \Delta g_{m-1}$. Thus, we wish to maximize Equation (4).

To create a move set that results in an effectively mixing chain, we consider which types of local moves result in jumps between posterior modes. As mentioned earlier, structures that differ by a single edge will probably have similar likelihoods. Additionally, structures that have slightly different edge change sets will have similar likelihoods. The *add edge*, *remove edge*, *add to edge set*, *remove from edge set*, and *move from edge set* moves are listed as $(M_1) - (M_5)$ in Table 1 in the Appendix.

Knowing in advance the times at which all the transitions occur is often unrealistic. When the number of transitions is known but the times are unknown *a priori* (KNUT setting), the transition times $T$ must also be estimated *a posteriori*.

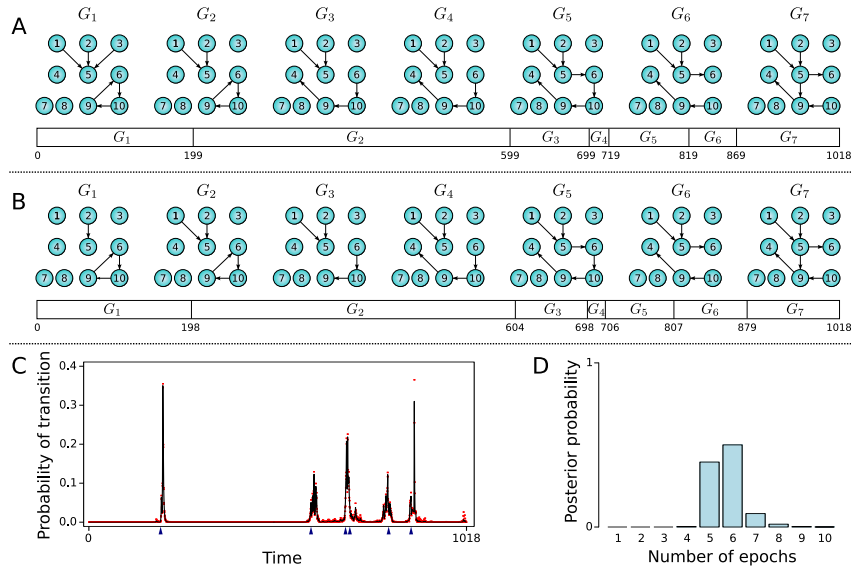

Figure 1: Structure learning of nsDBNs under several settings. **A.** True non-stationary data-generation process. Under the KNKT setting, the recovered structure is exactly this one. **B.** Under the KNUT setting, the algorithm learns the model-averaged nsDBN structure shown. **C:** Posterior probabilities of transition times when learning an nsDBN in the UNUT setting (with $\lambda_s = 1$ and $\lambda_m = 5$). The blue triangles represent the true transition times and the red dots represent one standard deviation from the mean probability obtained from several runs. **D:** Posterior probabilities of the number of epochs.

Structures with the same edge sets but slightly different transition times will probably have similar likelihoods. Therefore, we can add a new move that proposes a local shift to one of the transition times: let $d$ be some small positive integer and let the new time $t_i'$ be drawn from a discrete uniform distribution $t_i' \sim DU(t_i - d, t_i + d)$ with the constraint that $t_{i-1} < t_i' < t_{i+1}$. Initially, we set the $m - 1$ transition times so that the epochs are roughly equal in length. The complete move set for this setting includes all of the moves described previously as well as the new *local shift move*, listed as $(M_6)$ in Table 1 in the Appendix.

Finally, when the number and times of transitions are unknown (UNUT setting), both $m$ and $T$ must be estimated. While this is the most interesting setting, it is also the most difficult since one of the unknowns is the number of unknowns. Using the reversible jump Markov chain Monte Carlo sampling technique [11], we can further augment the move set to allow for the number of transitions to change. Since the number of epochs $m$ is allowed to vary, this is the only setting that incorporates the prior on $m$.

To allow the number of transitions to change during sampling, we introduce *merge* and *split* operations to the move set. For the merge operation, two adjacent edge sets ($\Delta g_i$ and $\Delta g_{i+1}$) are combined to create a new edge set. The transition time of the new edge set is selected to be the mean of the previous locations weighted by the size of each edge set: $t_i' = (S_i t_i + S_{i+1} t_{i+1})/(S_i + S_{i+1})$. For the split operation, an edge set $\Delta g_i$ is randomly chosen and randomly partitioned into two new edge sets $\Delta g_i'$ and $\Delta g_{i+1}'$ with all subsequent edge sets re-indexed appropriately. Each new transition time is selected as described above. The move set is completed with the inclusion of the *add transition time* and *delete transition time* operations. These moves are similar to the split and merge operations except they also increase or decrease $s$, the total number of edge changes in the structure. The four additional moves are listed as $(M_7) - (M_{10})$ in Table 1 in the Appendix.

## 5 Results on simulated data

To evaluate the effectiveness of our method, we first apply it to a small, simulated dataset. The first experiment is on a simulated ten node network with six single-edge changes between seven

epochs where the length of each epoch varies between 20 and 400 observations. The true network is shown in Figure 1A. For each of the three settings, we generate ten individual datasets and then collect 250,000 samples from each, with the first 50,000 samples thrown out for burn-in. We repeat the sample collection 25 times for each dataset to obtain variance estimates on posterior quantities of interest. The sample collection takes about 25 seconds for each dataset on a 3.6GHz dual-core Intel Xeon machine with 4 GB of RAM, but all runs can easily be executed in parallel. To obtain a consensus (model averaged) structure prediction, an edge is considered present at a particular time if the posterior probability of the edge is greater than 0.5.

In the KNKT setting, the sampler rapidly converges to the correct solution. The value of $\lambda_m$ has no effect in this setting, and the value of $\lambda_s$ is varied between 0.1 and 50. The predicted structure is identical to the true structure shown in Figure 1A for a broad range of values: $0.5 \leq \lambda_s \leq 10.0$, indicating robust and accurate learning.

In the KNUT setting, transition times are unknown and must be estimated *a posteriori*. The value of $\lambda_m$ still has no effect in this setting and the value of $\lambda_s$ is again varied between 0.1 and 50. The predicted consensus structure is shown in Figure 1B for $\lambda_s = 5.0$; this choice of $\lambda_s$ provides the most accurate predictions.

The estimated structure and transition times are very close to the truth. All edges are correct, with the exception of two missing edges in $G_1$, and the predicted transition times are all within 10 of the true transition times. We also discovered that the convergence rate under the KNUT and the KNKT settings were very similar for a given $m$. This implies that the posterior over transition times is quite smooth; therefore, the mixing rate is not greatly affected when sampling transition times. Finally, we consider the UNUT setting, when the number and times of transitions are both unknown.

We use the range $1 \leq \lambda_s \leq 5$ because we know from the previous settings that the most accurate solutions were obtained from a prior within this range; the range $1 \leq \lambda_m \leq 50$ is selected to provide a wide range of estimates for the prior since we have no *a priori* knowledge of what it should be.

We can examine the posterior probabilities of transition times over all sampled structures, shown in Figure 1C. Highly probable transition times correspond closely with the true transition times indicated by blue triangles; nevertheless, some uncertainty exists on about the exact locations of $t_3$ and $t_4$ since the fourth epoch is exceedingly short. We can also examine the posterior number of epochs, shown in Figure 1D. The most probable posterior number of epochs is six, close to the true number of seven.

To identify the best parameter settings for $\lambda_s$ and $\lambda_m$, we examine the best F1-measure (the harmonic mean of the precision and recall) for each. The best F1-measure of 0.992 is obtained when $\lambda_s = 5$ and $\lambda_m = 1$, although nearly all choices result in an F1-measure above 0.90 (see Appendix).

To evaluate the scalability of our technique, we also simulated data from a 100 variable network with an average of fifty edges over five epochs spanning 4800 observations, with one to three edges changing between each epoch. Learning nsDBNs on this data for $\lambda_s \in \{1, 2, 5\}$ and $\lambda_m \in \{2, 3, 5\}$ results in F1-measures above 0.93, with the $\lambda_s = 1$ and $\lambda_m = 5$ assignments to be best for this data, with an F1-measure of 0.953.

# 6 Results on *Drosophila* muscle development gene expression data

We also apply our method to identify non-stationary networks using *Drosophila* development gene expression data from [12]. This data contains expression measurements over 66 time steps of 4028 *Drosophila* genes throughout development and growth during the embryonic, larval, pupal, and adult stages of life. Using a subset of the genes involved in muscle development, some researchers have identified a single directed network [13], while others have learned a time-varying undirected network [4]. To facilitate comparison with as many existing methods as possible, we apply our method to the same data. Unfortunately, no other techniques predict non-stationary directed networks, so our prediction in Figure 2C is compared to the stationary directed network in Figure 2A and the non-stationary undirected network in Figure 2B.

While all three predictions share many edges, certain similarities between our prediction and one or both of the other two predictions are of special interest. In all three predictions, a cluster seems to form around *myo61f*, *msp-300*, *up*, *mhc*, *prm*, and *mlc1*. All of these genes except *up* are in the

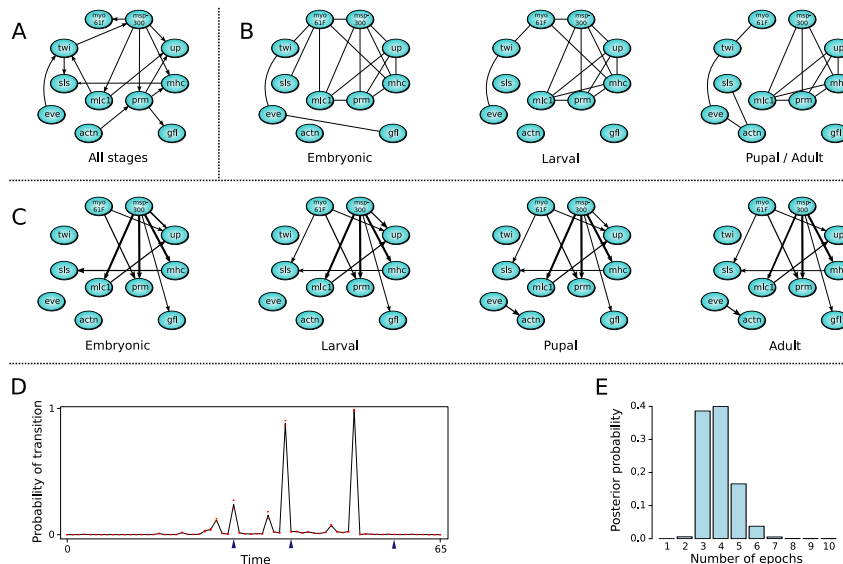

Figure 2: Learning nsDBNS from the *Drosophila* muscle development data. **A.** The directed network reported by [13]. **B.** The undirected networks reported by [4]. **C.** The nsDBN structure learned under the KNKT setting with $\lambda_s = 2.0$. Only the edges that occurred in greater than 50 percent of the samples are shown, with thicker edges representing connections that occurred more frequently. **D.** Posterior probabilities of transition times using $\lambda_m = \lambda_s = 2$ under the UNUT setting. Blue triangles represent the borders of embryonic, larval, pupal, and adult stages. **E.** Posterior probability of the number of epochs under the UNUT setting.

*myosin* family, which contains genes involved in muscle contraction. Within the directed predictions, *msp-300* primarily serves as a hub gene that regulates the other *myosin* family genes. It is interesting to note that the undirected method predicts connections between *mcl1*, *prm*, and *mhc* while neither directed method make these predictions. Since *msp-300* seems to serve as a regulator to these genes, the method from [4] may be unable to distinguish between direct interactions and correlations due to its undirected nature.

Despite the similarities, some notable differences exist between our prediction and the other two predictions. First, we predict interactions from *myo61f* to both *prm* and *up*, neither of which is predicted in the other methods, suggesting a greater role for *myo61f* during muscle development. Also, we do not predict any interactions with *twi*. During muscle development in *Drosophila*, *twi* acts as a regulator of *mef2* that in turn regulates some *myosin* family genes, including *mlc1* and *mhc* [14]; our prediction of no direct connection from *twi* mirrors this biological behavior. Finally, we note that in our predicted structure, *actn* never connects as a regulator (parent) to any other genes, unlike in the network in Figure 2A. Since *actn* (*actinin*) only binds *actin*, we do not expect it to regulate other muscle development genes, even indirectly.

We can also look at the posterior probabilities of transition times and epochs under the UNUT setting. These plots are shown in Figure 2D and 2E, respectively. The transition times with high posterior probabilities correspond well to the embryonic→larval and the larval→pupal transitions, but a posterior peak occurs well before the supposed time of the pupal→adult transition; this reveals that the gene expression program governing the transition to adult morphology is active well before the fly emerges from the pupa, as would clearly be expected. Also, we see that the most probable number of epochs is three or four, mirroring closely the total number of developmental stages.

Since we could not biologically validate the fly network, we generated a non-stationary time-series with the same number of nodes and a similar level of connectivity to evaluate the accuracy a recovered nsDBN on a problem of exactly this size. We generated data from an nsDBN with 66 observations and transition times at 30, 40, and 58 to mirror the number of observations in embryonic, larval, pupal, and adult stages of the experimental fly data. Since it is difficult to estimate the amount of noise in the experimental data, we simulated noise at 1:1 to 4:1 signal-to-noise ratios.

Finally, since many biological processes have more variables than observations, we examined the effect of increasing the number of experimental replicates. We found that the best F1-measures (greater than 0.75 across all signal-to-noise ratios and experimental replicates) were obtained when $\lambda_m = \lambda_s = 2$, which is why we used those values to analyze the *Drosophila* muscle network data.

## 7 Discussion

Non-stationary dynamic Bayesian networks provide a useful framework for learning Bayesian networks when the generating processes are non-stationary. Using the move sets described in this paper, nsDBN learning is efficient even for networks of 100 variables, generalizable to situations of varying uncertainty (KNKT, KNUT, and UNUT), and the predictions are stable over many choices of hyper-parameters. Additionally, by using a sampling-based approach, our method allows us to assess a confidence for each predicted edge—an advantage that neither [13] nor [4] share.

We have demonstrated the feasibility of learning an nsDBN in all three settings using simulated data, and in the KNKT and UNUT settings using real biological data. Although the predicted fly muscle development networks are difficult to verify, simulated experiments of a similar scale demonstrate highly accurate predictions, even with noisy data and few replicates.

Non-stationary DBNs offer all of the advantages of DBNs (identifying directed non-linear interactions between multivariate time-series) and are additionally able to identify non-stationarities in the interactions between time-series. In future work, we hope to analyze data from other fields that have traditionally used dynamic Bayesian networks and instead use nsDBNs to identify and model previously unknown or uncharacterized non-stationary behavior.

## References

[1] Nir Friedman, Michal Linial, Iftach Nachman, and Dana Pe'er. Using Bayesian networks to analyze expression data. In *RECOMB 4*, pages 127–135. ACM Press, 2000.

[2] V. Anne Smith, Jing Yu, Tom V. Smulders, Alexander J. Hartemink, and Erich D. Jarvis. Computational inference of neural information flow networks. *PLoS Computational Biology*, 2(11):1436–1449, 2006.

[3] Steve Hanneke and Eric P. Xing. Discrete temporal models of social networks. In *Workshop on Statistical Network Analysis, ICML 23*, 2006.

[4] Fan Guo, Steve Hanneke, Wenjie Fu, and Eric P. Xing. Recovering temporally rewiring networks: A model-based approach. In *ICML 24*, 2007.

[5] Makram Talih and Nicolas Hengartner. Structural learning with time-varying components: Tracking the cross-section of financial time series. *Journal of the Royal Statistical Society B*, 67(3):321–341, 2005.

[6] Xiang Xuan and Kevin Murphy. Modeling changing dependency structure in multivariate time series. In *ICML 24*, 2007.

[7] David Heckerman, Dan Geiger, and David Maxwell Chickering. Learning Bayesian networks: The combination of knowledge and statistical data. *Machine Learning*, 20(3):197–243, 1995.

[8] Claudia Tarantola. MCMC model determination for discrete graphical models. *Statistical Modelling*, 4(1):39–61, 2004.

[9] P Krause. Learning probabilistic networks. *The Knowledge Engineering Review*, 13(4):321–351, 1998.

[10] Kevin Murphy. Learning Bayesian network structure from sparse data sets. U.C. Berkeley Technical Report, Computer Science Department 990, University of California at Berkeley, 2001.

[11] Peter J. Green. Reversible jump Markov chain Monte Carlo computation and Bayesian model determination. *Biometrika*, 82(4):711–732, 1995.

[12] M Arbeitman, E Furlong, F Imam, E Johnson, B Null, B Baker, M Krasnow, M Scott, R Davis, and K White. Gene expression during the life cycle of *Drosophila melanogaster*. *Science*, 5590(297):2270–2275, 2002.

[13] Wentao Zhao, Erchin Serpedin, and Edward R. Dougherty. Inferring gene regulatory networks from time series data using the minimum description length principle. *Bioinformatics*, 22(17):2129–2135, 2006.

[14] T Sandmann, L Jensen, J Jakobsen, M Karzynski, M Eichenlaub, P Bork, and E Furlong. A temporal map of transcription factor activity: *mef2* directly regulates target genes at all stages of muscle development. *Developmental Cell*, 10(6):797–807, 2006.

